# Uniqueness of Belief Propagation on Signed Graphs

**Yusuke Watanabe**[*]
The Institute of Statistical Mathematics
10-3 Midori-cho, Tachikawa, Tokyo 190-8562, Japan
`watay@ism.ac.jp`

## Abstract

While loopy Belief Propagation (LBP) has been utilized in a wide variety of applications with empirical success, it comes with few theoretical guarantees. Especially, if the interactions of random variables in a graphical model are strong, the behaviors of the algorithm can be difficult to analyze due to underlying phase transitions. In this paper, we develop a novel approach to the uniqueness problem of the LBP fixed point; our new "necessary and sufficient" condition is stated in terms of graphs and signs, where the sign denotes the types (attractive/repulsive) of the interaction (i.e., compatibility function) on the edge. In all previous works, uniqueness is guaranteed only in the situations where the strength of the interactions are "sufficiently" small in certain senses. In contrast, our condition covers arbitrary strong interactions on the specified class of signed graphs. The result of this paper is based on the recent theoretical advance in the LBP algorithm; the connection with the graph zeta function.

## 1 Introduction

The belief propagation algorithm [1] was originally proposed as an efficient method for the exact computation in the inference with graphical models associated to trees; the algorithm has been extended to general graphs with cycles and called Loopy Belief Propagation (LBP) algorithm. It has shown empirical success in a wide class of problems including computer vision, compressed sensing and error correcting codes [2, 3, 4]. In such applications, existence of cycles and strong interactions between variables make the behaviors of the LBP algorithm difficult to analyze. In this paper we propose a novel approach to the uniqueness problem of LBP fixed point.

Although a considerable number of researches have been done in this decade [5, 6], understating of the LBP algorithm is not yet complete. An important step toward better understanding of the algorithm has been the variational interpretation by the Bethe free energy function; the fixed points of LBP correspond to the stationary points of the Bethe free energy function [7]. This view provides a number of algorithms that (provably) find a stationary point of the Bethe free energy function [8, 9, 10, 11]. For the uniqueness problem of the LBP fixed point a number of conditions has been proposed [12, 13, 14, 15]. (Note that the convergence property implies uniqueness by definition.) In all previous works, the uniqueness is guaranteed only in the situations where the strength of the interactions are "sufficiently" small in certain senses.

In this paper we propose a completely new approach to the uniqueness condition of the LBP algorithm; it should be emphasized that strength of interactions on specified class of signed graphs can be arbitrary large in this condition. (The signs denote the attractive/repulsive types of the compatibility function on the edges.) Generally speaking, the behavior of the algorithm is complex if the strength of interactions are strong. In such regions, phase transition phenomena can occur in the underlying computation tree [15], making theoretical analyses difficult. To overcome such difficulties,

---

[*]Current affiliation: SONY, Intelligent Systems Research Laboratory. YusukeB.Watanabe@jp.sony.com

we utilize the connection between the Bethe free energy and the graph zeta function established in [16]; the determinant of the Hessian of the Bethe free energy equals the reciprocal of the graph zeta function up to a positive factor. Combined with the index formula [16], the uniqueness problem is reduced to a positivity property of the graph zeta function.

This paper is organized as follows. In section 2 we introduce the background of LBP. In section 3 we explain the condition for the uniqueness, which is the main result of this paper. In section 4 the proof of the main result is given by a graph theoretic approach. In section 5 we remark foregoing researches based on the new technique.

## 2 Loopy Belief Propagation, Bethe free energy and graph zeta function

In this section, we provide basic facts on LBP; the connection with the Bethe free energy and graph zeta function. Throughout this paper, $G = (V, E)$ is a connected undirected graph with $V$, the vertices, and $E$, the undirected edges. We consider the binary pairwise model, which is given by the following factorization form with respect to $G$:

$$p(\boldsymbol{x}) = \frac{1}{Z} \prod_{ij \in E} \psi_{ij}(x_i, x_j) \prod_{i \in V} \psi_i(x_i), \tag{1}$$

where $\boldsymbol{x} = (x_i)_{i \in V}$ is a list of binary ( *i.e.*, $x_i \in \{\pm 1\}$) variables, $Z$ is the normalization constant and $\psi_{ij}, \psi_i$ are positive functions called *compatibility functions*. Without loss of generality we assume that $\psi_{ij}(x_i, x_j) = \exp(J_{ij} x_i x_j)$ and $\psi_i(x_i) = \exp(h_i x_i)$. We refer $J_{ij}$ as *interaction* and its absolute value as "strength".

In various applications, we would like to compute marginal distributions

$$p_i(x_i) := \sum_{\boldsymbol{x} \backslash \{x_i\}} p(\boldsymbol{x}) \qquad \text{and} \qquad p_{ij}(x_i, x_j) := \sum_{\boldsymbol{x} \backslash \{x_i x_j\}} p(\boldsymbol{x}) \tag{2}$$

though exact computations are often intractable due to the combinatorial complexities. If the graph is a tree, however, they are efficiently computed by the belief propagation algorithm [1]. Even if the graph has cycles, the direct application of the algorithm (Loopy Belief Propagation; LBP) often gives good approximation [6].

LBP is a message passing algorithm. For each directed edge, a message vector $\mu_{i \to j}(x_j)$ is assigned and initialized arbitrarily. The update rule of messages is given by

$$\mu_{i \to j}^{new}(x_j) \propto \sum_{x_i} \psi_{ji}(x_j, x_i) \psi_i(x_i) \prod_{k \in N_i \backslash j} \mu_{k \to i}(x_i), \tag{3}$$

where $N_i$ is the neighborhood of $i \in V$. The order of edges in the update is arbitrary; the set of fixed point does not depend on the order. If the messages converge to some fixed point $\{\mu_{i \to j}^\infty(x_j)\}$, the approximations of $p_i(x_i)$ and $p_{ij}(x_i, x_j)$ are calculated as

$$b_i(x_i) \propto \psi_i(x_i) \prod_{k \in N_i} \mu_{k \to i}^\infty(x_i), \tag{4}$$

$$b_{ij}(x_i, x_j) \propto \psi_{ij}(x_i, x_j) \psi_i(x_i) \psi_j(x_j) \prod_{k \in N_i \backslash j} \mu_{k \to i}^\infty(x_i) \prod_{k \in N_j \backslash i} \mu_{k \to j}^\infty(x_j), \tag{5}$$

with normalization $\sum_{x_i} b_i(x_i) = 1$ and $\sum_{x_i, x_j} b_{ij}(x_i, x_j) = 1$. From (3) and (5), the constraints $b_{ij}(x_i, x_j) > 0$, and $\sum_{x_j} b_{ij}(x_i, x_j) = b_i(x_i)$ are automatically satisfied.

### 2.1 The Bethe free energy

The LBP algorithm is interpreted as a variational problem of the Bethe free energy function [7]. In this formulation, the domain of the function is given by

$$L(G) = \left\{ \{q_i, q_{ij}\}; q_{ij}(x_i, x_j) > 0, \sum_{x_i, x_j} q_{ij}(x_i, x_j) = 1, \sum_{x_j} q_{ij}(x_i, x_j) = q_i(x_i) \right\} \tag{6}$$

and element of this set is called pseudomarginals, i.e., a set of locally consistent probability distributions. The closure of this set is called *local marginal polytope* [6]. The objective function called *Bethe free energy* is defined on $L(G)$ by:

$$F(q) := - \sum_{ij \in E} \sum_{x_i x_j} q_{ij}(x_i, x_j) \log \psi_{ij}(x_i, x_j) - \sum_{i \in V} \sum_{x_i} q_i(x_i) \log \psi_i(x_i)$$
$$+ \sum_{ij \in E} \sum_{x_i x_j} q_{ij}(x_i, x_j) \log q_{ij}(x_i, x_j) + \sum_{i \in V} (1 - d_i) \sum_{x_i} q_i(x_i) \log q_i(x_i), \qquad (7)$$

where $d_i = |N_i|$. The outcome of this variational problem is the same as that of LBP. More precisely, there is a one-to-one correspondence between the set of stationary points of the Bethe free energy and the set of fixed points of LBP. The correspondence is given by (4, 5).

## 2.2  Zeta function and Ihara's formula

In this section, we explain the connection of LBP to the graph zeta function. We use the following terms for graphs [17, 16]. Let $\vec{E}$ be the set of directed edges obtained by duplicating undirected edges. For each directed edge $e \in \vec{E}$, $o(e) \in V$ is the *origin* of $e$ and $t(e) \in V$ is the *terminus* of $e$. For $e \in \vec{E}$, the *inverse edge* is denoted by $\bar{e}$, and the corresponding undirected edge by $[e] = [\bar{e}] \in E$. A *closed geodesic* in $G$ is a sequence $(e_1, \ldots, e_k)$ of directed edges such that $t(e_i) = o(e_{i+1}), e_i \neq \bar{e}_{i+1}$ for $i \in \mathbb{Z}/k\mathbb{Z}$. For a closed geodesic $c$, we may form the *m-multiple*, $c^m$, by repeating it $m$-times. A closed geodesic $c$ is *prime* if there are no closed geodesic $d$ and natural number $m(\geq 2)$ such that $c = d^m$. For example, a closed geodesic $c = (e_1, e_2, e_3, e_1, e_2, e_3)$ is not prime and $c = (e_1, e_2, e_3, e_4, e_1, e_2, e_3)$ is prime. Two closed geodesics are said to be *equivalent* if one is obtained by cyclic permutation of the other. For example, closed geodesics $(e_1, e_2, e_3), (e_2, e_3, e_1)$ and $(e_3, e_1, e_2)$ are equivalent. An equivalence class of prime closed geodesics is called a *prime cycle*. Let $P$ be the set of prime cycles of $G$. For given (complex or real) weights $\boldsymbol{u} = (u_e)_{e \in \vec{E}}$, the *Ihara's graph zeta function* [18, 19] is given by

$$\zeta_G(\boldsymbol{u}) := \prod_{\mathfrak{p} \in P} (1 - g(\mathfrak{p}))^{-1} \quad g(\mathfrak{p}) := u_{e_1} \cdots u_{e_k} \quad \text{for } \mathfrak{p} = (e_1, \ldots, e_k),$$
$$= \det(I - \mathcal{U}\mathcal{M})^{-1},$$

where the second equality is the determinant representation [19] with matrices indexed by the directed edges. The definitions of $\mathcal{M}$ and $\mathcal{U}$ are

$$\mathcal{M}_{e,e'} := \begin{cases} 1 & \text{if } e \neq \bar{e}' \text{ and } o(e) = t(e'), \\ 0 & \text{otherwise.} \end{cases} \qquad (8)$$

and $\mathcal{U}_{e,e'} := u_e \delta_{e,e'}$, respectively.

The following theorem gives the connection between the Bethe free energy and the zeta function. More precisely, the theorem asserts that the determinant of the Hessian of the Bethe free energy function is the reciprocal of the zeta function up to a positive factor.

**Theorem 1** ([16, 20]). *The following equality holds at any point of $L(G)$:*

$$\zeta_G(\boldsymbol{u})^{-1} = \det(\nabla^2 F) \prod_{ij \in E} \prod_{x_i, x_j = \pm 1} q_{ij}(x_i, x_j) \prod_{i \in V} \prod_{x_i = \pm 1} q_i(x_i)^{1 - d_i} 2^{2|V| + 4|E|} \qquad (9)$$

*where the derivatives are taken over a affine coordinate of $L(G)$: $m_i = \mathrm{E}_{q_i}[x_i], \chi_{ij} = \mathrm{E}_{q_{ij}}[x_i x_j]$, and*

$$u_{i \to j} = \frac{\chi_{ij} - m_i m_j}{\{(1 - m_i^2)(1 - m_j^2)\}^{1/2}} = \frac{\mathrm{Cov}_{q_{ij}}[x_i, x_j]}{\{\mathrm{Var}_{q_i}[x_i] \mathrm{Var}_{q_j}[x_j]\}^{1/2}} . =: \beta_{ij} \qquad (10)$$

Note that, from (7), the Hessian $\nabla^2 F$ does not depend on $J_{ij}$ and $h_i$. Since the weight (10) in Theorem 1 is symmetric with respect to the inversion of edges, the zeta function can be reduced to undirected edge weights. To avoid confusion, we introduce a notation: the zeta function of undirected edge weights $\boldsymbol{\beta} = (\beta_{ij})_{ij \in E}$ is denoted by $Z_G(\boldsymbol{\beta})$. Note also that, since $\beta_{ij}$ is the correlation coefficient of $q_{ij}$, we have $|\beta_{ij}| < 1$. The equality does not occur by the positivity assumption of probabilities.

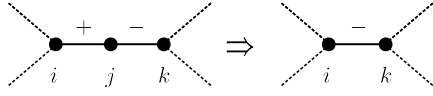

Figure 1: w1-reduction

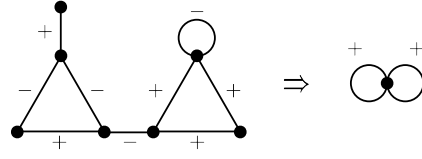

Figure 2: Example of the complete w-reduction.

# 3 Signed graphs with unique solution

In this section, we state the main result of this paper, Theorem 3. The result shows a new type of approach towards uniqueness conditions. The proof of the theorem is given in the next section.

## 3.1 Existing conditions on uniqueness

There have been many works on the uniqueness and/or convergence of the LBP algorithm for discrete graphical models [12, 13, 14, 15] and Gaussian graphical models [21]. As we are discussing binary pairwise graphical models, we review some of the conditions for the model. The following condition is given by Mooij and Kappen:

**Theorem 2.** *[[13]] Let $\rho(X)$ denote the spectral radius (i.e., the maximum of the absolute value of the eigenvalues) of a matrix $X$. If $\rho(\mathcal{J}\mathcal{M}) < 1$, then the LBP converges to the unique fixed point, where $\mathcal{J}$ is a diagonal matrix defined by $\mathcal{J}_{e,e'} = \tanh(|J_e|)\delta_{e,e'}$.*

This theorem gives the uniqueness property by bounding the strengths of the interactions, i.e., $\{|J_{ij}|\}_{ij \in E}$. Therefore, the condition does not depend on the signs of the interactions. The situations are the same in other existing conditions [12, 13, 14, 15]. For example, Heskes's condition [12] is

$$\sum_{j \in N_i} |J_{ij}| < 1. \tag{11}$$

These conditions are unsatisfactory in a sense that they do not use the information of the signs, $\{\operatorname{sgn} J_{ij}\}_{ij \in E}$. In fact, the behaviors of LBP algorithm can be dramatically different if the signs of the compatibility functions are changed. Note that each edge compatibility function $\psi_{ij}$ tend to force the variables $x_i, x_j$ equal if $J_{ij} > 0$ and not equal if $J_{ij} < 0$; the first case is called *attractive* interaction and the latter *repulsive*. In contrast to the above uniqueness conditions, we pursue another approach: we use the information of signs, $\{\operatorname{sgn} J_{ij}\}_{ij \in E}$, rather than the strengths. In this paper, we characterize the signed graphs that guarantee the uniqueness of the solution; this result is stated in Theorem 3.

## 3.2 Statement of main theorem of this section

We introduce basic terms to state the main theorem. A *signed graph*, $(G, s)$, is a graph equipped with a sign map, $s$, from the edges to $\{\pm 1\}$. A compatibility function defines the sign function, $s$, by $s(ij) = \operatorname{sgn} J_{ij}$. The sign function of all plus (resp. minus) sign is denoted by $s_+$ (resp. $s_-$). The deletion and subgraph of a signed graph is defined naturally restricting the sign function.

**Definition 1.** A *w-reduction* of a signed graph $(G, s)$ is a signed graph that is obtained by one of the following operations:

**(w1)** Erasure of a vertex of degree two. (Let $j$ be a vertex of degree two and $ij, jk$ ($i \neq k$) be the connecting edges. Delete them and make a new edge $ik$ with the sign $s(ij)s(jk)$. See Figure 1.)

**(w2)** Deletion of a loop with minus sign. (An edge $ij$ is called a *loop* if $i = j$.)

**(w3)** Contraction of a bridge. (An edge is a *bridge* if the deletion of the edge makes the number of the connected component increase. The sign on the bridge can be either $+1$ or $-1$.)

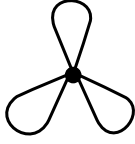 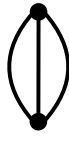 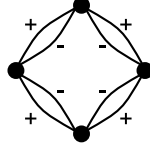 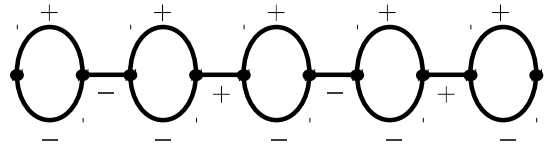

Figure 3: $B_3$    Figure 4: $P_3$    Figure 5: $D_4$.         Figure 6: Example 4 in Subsection 3.3.

Note that all the operations decrease the number of edges by one. A signed graph is *w-reduced* if no w-reduction is applicable. Any signed graph is reduced to the unique w-reduced signed graph called the *complete w-reduction*. Example of a complete w-reduction is given in Figure 2. From the viewpoint of the computational complexity, finding the complete w-reduction is easy. (See the supplementary material for further discussions.)

Here are important (signed) graphs. See Figures 3, 4 and 5. A bouquet graph, $B_n$, is a graph with the single node with $n$ loops. $P_n$ is a graph with two vertices and $n$ parallel edges. $K_n$ is the complete graph of $n$ vertices. $C_n$ is cycle of length $n$. $D_n$ is a signed graph obtained by duplicating each edge of $C_n$ with plus and minus signs.

**Definition 2.** Two signed graphs $(G, s)$ and $(G, s')$ are said to be *gauge equivalent* if there exists a map $g : V \longrightarrow \{\pm 1\}$ such that $s'(ij) = s(ij)g(i)g(j)$. The map $g$ is called *gauge transformation*.

**Theorem 3.** *For a signed graph $(G, s)$ the following conditions are equivalent.*

1. *LBP algorithm on G has the unique fixed point for any compatibility functions with sign s.*

2. *The complete w-reduction of $(G, s)$ is one of the followings:* (i) $B_0$ (ii) $(B_1, +)$ (iii) $(P_3, +, -, -)$ *and* $(P_3, +, +, -)$. (iv) $(K_4, s_-)$ *and its gauge equivalent signed graphs.* (v) $D_n$ *and its w-reduced subgraphs* $(n \geq 2)$.

The proof of this theorem is given in the next section.

## 3.3 Examples and experiments

In this subsection we present concrete examples of signed graphs which do or do not satisfy the condition of Theorem 3.

(Ex.1) Trees and graphs with a single cycle: In these cases it is well known that LBP has the unique fixed point irrespective of the compatibility functions [1, 22]. This fact is easily derived by Theorem 3 since the complete w-reduction of them are $B_0$ or $(B_1, +)$. (Ex.2) Complete graph $K_n$: $(K_n, s)$ is w-reduced as we can not apply w-reduction. For $n = 4$, the condition of sign is given in 2.(iv). If $n \geq 5$ it does not satisfy the condition for any sign. (Ex.3) $2 \times 2$ grid graph: This graph does not satisfy the condition for any sign because its complete w-reduction is different from the signed graphs in the item 2 of Theorem 3. (Ex.4) Consider a signed graph in Figure 6. Notice that the products of signs along the five cycles are all minus. Applying (w2) and (w3), we see that the complete w-reduction is $B_0$. Therefore the signed graph satisfies the condition.

We experimentally check convergence behaviors of the LBP algorithm on $D_4$, which satisfies the condition of Theorem 3. Since the LBP fixed point is unique, it is the absolute minimum of the Bethe free energy function. We set the compatibility functions $J_{ij} = \pm J, h_i = h$ and initialized messages randomly. We judged convergence if average message update is less than $10^{-3}$ after 50 iterations. The result is shown in Figure 7. LBP is not convergent in the right white region and convergent in the rest of gray region. Convergence is theoretically guaranteed for $\tanh(|J|) < 1/3$ ($|J| \lessapprox 0.347$) by Theorem 2. In the non-convergent region LBP appears to be unstable around the fixed point.

## 4 Proofs: conditions in terms of graph zeta function

The aim of this subsection is to prove Theorem 3. For the proof, Lemma 2, which is purely a result of the graph zeta function, is utilized.

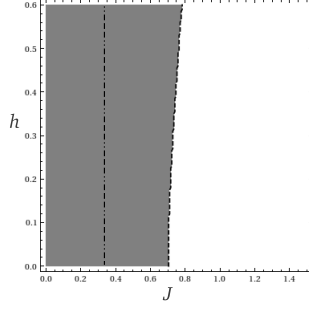

Figure 7: Convergence region of LBP.

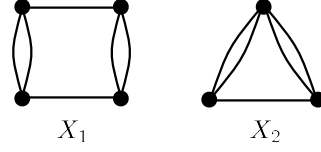

Figure 8: $X_1$ and $X_2$.

## 4.1 Graph theoretic results

We denote by $G - \epsilon$ the deletion of an undirected edge $\epsilon$ from a graph $G$ and by $G/\epsilon$ the contraction. A *minor* of a graph is obtained by the repeated applications of the deletion, contraction and removal of isolated vertices. The Deletion and contraction operations have natural meaning in the context of the graph zeta function as follows:

**Lemma 1.**

1. *Let $ij$ be an edge, then $\zeta_{G-ij}^{-1}(\boldsymbol{u}) = \zeta_G^{-1}(\tilde{\boldsymbol{u}})$, where $\tilde{u}_e$ is equal to $u_e$ if $[e] \neq ij$ and $0$ otherwise.*

2. *Let $ij$ be a non-loop edge, then $\zeta_{G/ij}^{-1}(\boldsymbol{u}) = \zeta_G^{-1}(\tilde{\boldsymbol{u}})$, where $\tilde{u}_e$ is equal to $u_e$ if $[e] \neq ij$ and $1$ otherwise.*

*Proof.* From the prime cycle representation of zeta functions, both of the assertions are trivial. $\quad\square$

Next, to prove Theorem 3, we formally define the notion of deletions, contractions and minors on signed graphs [23]. For a signed graph the *signed-deletion* of an edge is just the deletion of the edge along with the sign on it. The *signed-contraction* of a non-loop edge $ij \in E$ is defined up to gauge equivalence as follows. For any non-loop edge $ij$, there is a gauge equivalent signed graph that has the sign $+$ on $ij$. The signed-contraction is obtained by contracting the edge. The resulting signed graph is determined up to gauge equivalence. A *signed minor* of a signed graph is obtained by repeated applications of the signed-deletion, signed-contraction, and removal of isolated vertices.

**Lemma 2.** *For a signed graph, $(G, s)$, the following conditions are equivalent.*

1. *$(G, s)$ is U-type. That is, if $\beta_{ij} \in I_{s(ij)}$ for all $ij \in E$ then $Z_G^{-1}(\boldsymbol{\beta}) > 0$, where $\boldsymbol{\beta} = (\beta_{ij})_{ij \in E}$, $I_+ = [0, 1)$ and $I_- = (-1, 0]$.*

2. *$(G, s)$ is weakly U-type. That is, if $\beta_{ij} \in I_{s(ij)}$ for all $ij \in E$ then $Z_G^{-1}(\boldsymbol{\beta}) \geq 0$*

3. *$(B_2, s_+)$ is not contained as a signed minor.*

4. *The complete w-reduction of $(G, s)$ is one of the followings: (i) $B_0$ (ii) $(B_1, s_+)$ (iii) $(P_3, +, -, -)$ and $(P_3, +, +, -)$. (iv) $(K_4, s_-)$ and its gauge equivalent signed graphs. (v) $D_n$ and its w-reduced subgraphs $(n \geq 2)$.*

The uniqueness condition in Theorem 3 is equivalent to all the conditions in this lemma. Here, we remark properties of this condition (the proof is straightforward from definition and Lemma 2):

**(1)** $(G, s)$ is U-type iff its gauge equivalents are U-type.

**(2)** If $(G, s)$ is U-type then its signed minors are U-type.

We prove the equivalence cyclic manner. Here we give a sketch of the proof (Detail is given in the supplementary material.)

*Proof of* $1 \Rightarrow 2$. Trivial. □

*Proof of* $2 \Rightarrow 3$. If $(G, s)$ is weakly U-type, then its signed minors are weakly U-type; this is obvious from Lemma 1. However, direct computation of the zeta of $(B_2, s_+)$ shows that this signed graph is not weakly U-type. In fact, the directed edge matrix with weight of $B_2$ is

$$\mathcal{BM} = \begin{bmatrix} \beta_{\epsilon_1} & \beta_{\epsilon_1} & 0 & \beta_{\epsilon_1} \\ \beta_{\epsilon_2} & \beta_{\epsilon_2} & \beta_{\epsilon_2} & 0 \\ 0 & \beta_{\epsilon_1} & \beta_{\epsilon_1} & \beta_{\epsilon_1} \\ \beta_{\epsilon_2} & 0 & \beta_{\epsilon_2} & \beta_{\epsilon_2} \end{bmatrix}$$

and $\det(I - \mathcal{BM}) = (1 - \beta_{\epsilon_1})(1 - \beta_{\epsilon_2})(1 - \beta_{\epsilon_1} - \beta_{\epsilon_2} - 3\beta_{\epsilon_1}\beta_{\epsilon_2})$. This value can be negative in the region $0 \le \beta_{\epsilon_1}, \beta_{\epsilon_2} < 1$. □

*Proof of* $3 \Rightarrow 4$. Note that if $(G, s)$ does not contain $(B_2, s_+)$ as a signed minor then any w-reductions of $(G, s)$ also do not contain $(B_2, s_+)$ as a signed minor; we can check this property for each type of w-reductions, (w,1,2,3).

Therefore, it is sufficient to show that if a w-reduced signed graph $(G, s)$ does not contain $(B_2, +, +)$ as a signed minor then it is one of the five types. Notice that $G$ has no vertex of degree less than three. First, if the nullity of $G$ is less than three, it is not hard to see that the signed graph is type (i), (ii) or (iii). Secondly, we consider the case that the graph $G$ has nullity three. Note that all w-reduced signed graphs of nullity two have the signed minor $(B_1, +)$. Therefore, we can assume that $G$ does not have (plus) loop. Since $(G, s)$ is w-reduced, $G$ must be one of the following graphs: $K_4, P_4, X_1$ and $X_2$, where $X_1$ and $X_2$ are defined in Figure 8. It is easy to check that possible way of assigning signs on these graphs are one of the types, (iii-v). Finally, we consider the case of the nullity, $n$, is more than three. In this case, we can show that $(G, s)$ must be $D_n$ or its subgraph. (Details are found in the supplementary material.) □

*Proof of* $4 \Rightarrow 1$. First we claim the following statement: if

$$\zeta_G^{-1}(\boldsymbol{u}) \ge 0 \quad \forall \boldsymbol{u} = (u_e) \in \prod_{e \in \vec{E}} \{0, s([e])\}, \tag{12}$$

then $(G, s)$ is U-type. This claim can be proved using the property that $\zeta_G^{-1}(\boldsymbol{u}) = \det(I - \mathcal{UM})$ is linear for each variable, $u_e$. (That is, if we fix $\boldsymbol{u}$ except for one variable, say $u_{e_1}$, then $\zeta_G^{-1} = C_1 + C_2 u_{e_1}$.) Take the product of the closed intervals from $0$ to $s(e)$ ($e \in \vec{E}$) and make a hypercube. If there is a non-positive point in the hypercube then there must be a non-positive point in a face; we can repeat this argument until we arrive at a vertex.

We check the condition (12) for all the four classes. Notice that if $(G, s)$ satisfies (12) then its gauge equivalents, the deletion and signed-contraction has the same property. So far, we have proven the assertion for w-reduced graphs; we extend the proof to arbitrary signed graphs. For any signed graph, the complete w-reductions are obtained by first using reductions (w1,w2) and then reducing the bridges (w3) because (w3) always makes the degree bigger and does not make a loop. Therefore, the following two claims complete the proof.

**Claim 1.** *Let* $(G', s')$ *be a (w3)-reduction of a signed graph* $(G, s)$, *i.e., obtained by contraction of a bridge* $\epsilon$. *If* $(G', s')$ *has the property (12) then* $(G, s)$ *also has the property.*

*Proof of Claim 1.* Let $b$ and $\bar{b}$ be the corresponding directed edges of $\epsilon$. Since any prime cycles pass $b$ and $\bar{b}$ at the same number of times,

$$\zeta_G^{-1}(\boldsymbol{u}) = \zeta_{G-\epsilon}^{-1}(\tilde{\boldsymbol{u}}) + u_b u_{\bar{b}} f(\tilde{\boldsymbol{u}}), \tag{13}$$

where $\tilde{\boldsymbol{u}}$ is restriction of $\boldsymbol{u}$ on $G - \epsilon$ and $f$ is a function. Assume that $s(\epsilon) = 1$. (The case $s(\epsilon) = -1$ is completely analogous.) Since $(G', s')$ has the property (12), $(G, s)$ has the property for $(u_b, u_{\bar{b}}) = (1, 1)$. For $(u_b, u_{\bar{b}}) = (0, 0), (1, 0), (0, 1)$ cases, we can deduce form the property of $G - \epsilon$. ∎

**Claim 2.** *Let* $(G', s')$ *be a (w1) or (w2)-reduction of a signed graph* $(G, s)$. *If* $(G', s')$ *is U-type then* $(G, s)$ *is U-type.*

*Proof of Claim 2.* The case of (w1) is trivial. We prove the case (w2). From the multivariate Ihara's formula, the positivity of $Z_{G'}^{-1}(\boldsymbol{\beta})$ on the set $\prod_{ij \in E} I_{s(ij)}$ implies the positive definiteness of $I + \hat{\mathcal{D}}' - \hat{\mathcal{A}}'$ on the set. Adding a minus loop correspond to adding $2\beta^2(1-\beta^2)^{-1} - 2\beta(1-\beta^2)$ $= -2\beta(1+\beta)$ on the diagonal, where $-1 < \beta \le 0$. Therefore the new matrix is also positive definite and $(G,s)$ is U-type. ∎

□

## 4.2 Proof of Theorem 3

*Proof of 2 ⇒ 1.* The basic strategy is to use the following theorem.

**Theorem 4** (Index sum theorem [16]). *As usual, consider the Bethe free energy function, F, defined on $L(G)$. Assume that $\det \nabla^2 F(q) \ne 0$ for all LBP fixed points q. Then the sum of indices at the LBP fixed points are equal to one:*

$$\sum_{q: \nabla F(q) = 0} \operatorname{sgn}\left(\det \nabla^2 F(q)\right) = 1, \quad \text{where } \operatorname{sgn}(x) := \begin{cases} 1 & \text{if } x > 0, \\ -1 & \text{if } x < 0. \end{cases}$$

*(We call each summand, which is $+1$ or $-1$, the index of F at q.)*

At each LBP fixed point, the beta values for a solution can be computed using (10). Since the signs of $\beta_{ij}$ and $J_{ij}$ are equal [16], $\boldsymbol{\beta} = (\beta_{ij}) \in \prod_{ij \in E} I_{s(ij)}$ is satisfied. Therefore, from the assumption and Lemma 2, the index of the solution is positive. We conclude the uniqueness of the solution from the above index sum theorem. □

*Proof of 1 ⇒ 2.* We show the contraposition. From Lemma 2, $(G,s)$ is not weakly U-type; there is $\boldsymbol{\beta} = (\beta_{ij}) \in \prod_{ij \in E} I_{s(ij)}$ such that $\zeta_G^{-1}(\boldsymbol{\beta}) < 0$. Take pseudomarginals $q = \{q_{ij}\}_{ij \in E} \cup \{q_i\}_{i \in V}$ that has the correlation coefficients of $q_{ij}$ equal to $\beta_{ij}$. (For example, set $\chi_{ij} = \beta_{ij}, m_i = 0$.) We can choose $J_{ij}$ and $h_i$ such that

$$\prod_{ij \in E} q_{ij}(x_i, x_j) \prod_{i \in V} q_i^{1-d_i}(x_i) \propto \exp\left(\sum_{ij \in E} J_{ij} x_i x_j + \sum_{i \in V} h_i x_i\right). \tag{14}$$

This construction implies that $q$ correspond to a LBP fixed point with compatibility functions $\{J_{ij}, h_i\}$. This solution has index -1 by definition. If this is the unique solution, it contradicts the index sum formula. Therefore, there must be other solutions. □

## 5 Concluding remarks

In this paper we have developed a new approach to the uniqueness problem of the LBP algorithm. As a result, we have obtained a new class of LBPs that are guaranteed to have the unique solution. The uniqueness problem is reduced to the properties of graph zeta functions, Lemma 2, using the indexed formula. In contrast to the existing conditions, our uniqueness guarantee includes graphical models with strong interactions. Though our result is shown in the case of binary pairwise models, the idea can be extended to factor graph models with many states. In fact, Theorem 1 has been extended to the general settings of the LBP algorithm on factor graphs [20].

One direction for the future research is to combine the information of the signs and strengths of the interactions to show the uniqueness. The uniqueness problem is reduced to the positivity of the graph zeta function on a restricted set, rather than the hypercube of size one. If we can check the positivity of graph zeta functions theoretically or algorithmically, the result can be used for a better guarantee of the uniqueness.

## References

[1] J. Pearl. *Probabilistic Reasoning in Intelligent Systems: Networks of Plausible Inference.* Morgan Kaufmann Publishers, San Mateo, CA, 1988.

[2] P.F. Felzenszwalb and D.P. Huttenlocher. Efficient belief propagation for early vision. *International journal of computer vision*, 70(1):41–54, 2006.

[3] D. Baron, S. Sarvotham, and R.G. Baraniuk. Bayesian compressive sensing via belief propagation. *Signal Processing, IEEE Transactions on*, 58(1):269–280, 2010.

[4] R.J. McEliece, D.J.C. MacKay, and J.F. Cheng. Turbo decoding as an instance of Pearl's "belief propagation" algorithm. *IEEE J. Sel. Areas Commun.*, 16(2):140–52, 1998.

[5] S. Ikeda, T. Tanaka, and S. Amari. Stochastic reasoning, free energy, and information geometry. *Neural Computation*, 16(9):1779–1810, 2004.

[6] M.J. Wainwright and M.I. Jordan. Graphical models, exponential families, and variational inference. *Foundations and Trends in Machine Learning*, 1(1-2):1–305, 2008.

[7] J.S. Yedidia, W.T. Freeman, and Y. Weiss. Generalized belief propagation. *Adv. in Neural Information Processing Systems*, 13:689–95, 2001.

[8] A.L. Yuille. CCCP algorithms to minimize the bethe and kikuchi free energies: Convergent alternatives to belief propagation. *Neural computation*, 14(7):1691–1722, 2002.

[9] A.L. Yuille and A. Rangarajan. The concave-convex procedure. *Neural Computation*, 15(4):915–936, 2003.

[10] Y.W. Teh, M. Welling, et al. The unified propagation and scaling algorithm. *Advances in neural information processing systems*, 2:953–960, 2002.

[11] T. Heskes. Convexity arguments for efficient minimization of the bethe and kikuchi free energies. *Journal of Artificial Intelligence Research*, 26(1):153–190, 2006.

[12] T. Heskes. On the uniqueness of loopy belief propagation fixed points. *Neural Computation*, 16(11):2379–2413, 2004.

[13] J. M. Mooij and H. J. Kappen. Sufficient Conditions for Convergence of the Sum-Product Algorithm. *IEEE Transactions on Information Theory*, 53(12):4422–4437, 2007.

[14] A.T. Ihler, JW Fisher, and A.S. Willsky. Loopy belief propagation: Convergence and effects of message errors. *Journal of Machine Learning Research*, 6(1):905–936, 2006.

[15] S. Tatikonda and M.I. Jordan. Loopy belief propagation and Gibbs measures. *Uncertainty in AI*, 18:493–500, 2002.

[16] Y. Watanabe and K. Fukumizu. Graph zeta function in the bethe free energy and loopy belief propagation. *Adv. in Neural Information Processing Systems*, 22:2017–2025, 2009.

[17] M. Kotani and T. Sunada. Zeta functions of finite graphs. *J. Math. Sci. Univ. Tokyo*, 7(1):7–25, 2000.

[18] K. Hashimoto. Zeta functions of finite graphs and representations of p-adic groups. *Automorphic forms and geometry of arithmetic varieties*, 15:211–280, 1989.

[19] H.M. Stark and A.A. Terras. Zeta functions of finite graphs and coverings. *Advances in Mathematics*, 121(1):124–165, 1996.

[20] Y. Watanabe and K. Fukumizu. Loopy belief propagation, Bethe free energy and graph zeta function. *arXiv:1103.0605*.

[21] D.M. Malioutov, J.K. Johnson, and A.S. Willsky. Walk-sums and belief propagation in Gaussian graphical models. *The Journal of Machine Learning Research*, 7:2064, 2006.

[22] Y. Weiss. Correctness of Local Probability Propagation in Graphical Models with Loops. *Neural Computation*, 12(1):1–41, 2000.

[23] Thomas Zaslavsky. Characterizations of signed graphs. *Journal of Graph Theory*, 5(4):401–406, 1981.

